# Restructuring Sparse High Dimensional Data for Effective Retrieval

**Charles Lee Isbell, Jr.**
AT&T Labs
180 Park Avenue Room A255
Florham Park, NJ 07932-0971

**Paul Viola**
Artificial Intelligence Laboratory
Massachusetts Institute of Technology
Cambridge, MA 02139

## Abstract

The task in text retrieval is to find the subset of a collection of documents relevant to a user's information request, usually expressed as a set of words. Classically, documents and queries are represented as vectors of word counts. In its simplest form, relevance is defined to be the dot product between a document and a query vector–a measure of the number of common terms. A central difficulty in text retrieval is that the presence or absence of a word is not sufficient to determine relevance to a query. Linear dimensionality reduction has been proposed as a technique for extracting underlying structure from the document collection. In some domains (such as vision) dimensionality reduction reduces computational complexity. In text retrieval it is more often used to improve retrieval performance. We propose an alternative and novel technique that produces *sparse* representations constructed from sets of highly-related words. Documents and queries are represented by their distance to these sets, and relevance is measured by the number of common clusters. This technique significantly improves retrieval performance, is efficient to compute and shares properties with the optimal linear projection operator and the *independent components* of documents.

## 1  Introduction

The task in text retrieval is to find the subset of a collection of documents relevant to a user's information request, usually expressed as a set of words. Naturally, we would like to apply techniques from natural language understanding to this problem. Unfortunately, the sheer size of the data to be represented makes this difficult. We wish to process tens or hundreds of thousands of documents, each of which may contain hundreds of thousands of different words. It is clear that any useful approach must be time and space efficient.

Following (Salton, 1971), we adopt a modified Vector Space Model (VSM) for document representation. A document is a vector where each dimension is a count of occurrences for a different word[1].

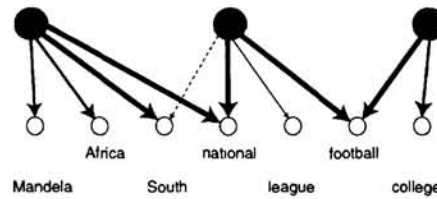

Figure 1: A Model of Word Generation. Independent topics give rise to specific words words according an unknown probability distribution (Line thickness indicates the likelihood of generating a word).

A collection of documents is a matrix, $D$, where each column is a document vector $d_i$. Queries are similarly represented.

We propose a topic based model for the generation of words in documents. Each document is generated by the interaction of a set of independent hidden random variables called *topics*. When a topic is active it causes words to appear in documents. Some words are very likely to be generated by a topic and others less so. Different topics may give rise to some of the same words. The final set of observed words results from a linear combination of topics. See Figure 1 for an example.

In this view of word generation, individual words are only weak indicators of underlying topics. Our task is to discover from data those collections of words that best predict the (unknown) underlying topics. The assumption that words are neither independent of one another or conditionally independent of topics motivates our belief that this is possible.

Our approach is to construct a set of linear operators which extract the independent topic structure of documents. We have explored different algorithms for discovering these operators include independent components analysis (Bell and Sejnowski, 1995). The inferred topics are then used to represent and compare documents.

Below we describe our approach and contrast it with Latent Semantic Indexing (LSI), a technique that also attempts to linearly transform the documents from "word space" into one more appropriate for comparison (Hull, 1994; Deerwester et al., 1990). We show that the LSI transformation has very different properties than the optimal linear transformation. We characterize some of these properties and derive an unsupervised method that searches for them. Finally, we present experiments demonstrating the robustness of this method and describe several computational and space advantages.

## 2 The Vector Space Model and Latent Semantic Indexing

The similarity between two documents using the VSM model is their inner product, $d_i^T d_j$. Queries are just short documents, so the relevance of documents to a query, $q$, is $D^T q$. There are several advantages to this approach beyond its mathematical simplicity. Above all, it is efficient to compute and store the word counts. While the word-document matrix has a very large number of potential entries, most documents do not contain very many of the possible words, so it is sparsely populated. Thus, algorithms for manipulating the matrix only require space and time proportional to the average number of different words that appear in a document, a number likely to be much smaller than the full dimensionality of the document matrix (in practice, non-zero elements represent about 2% of the total number of elements). Nevertheless, VSM makes an important tradeoff by sacrificing a great deal of document structure, losing context that may disambiguate meaning.

Any text retrieval system must overcome the fundamental difficulty that the presence or absence of a word is insufficient to determine relevance. This is due to two intrinsic problems of natural

---

(Frakes and Baeza-Yates, 1992). We incorporate these methods; however, such details are unimportant for this discussion.

language: *synonymy* and *polysemy*. Synonymy refers to the fact that a single underlying concept can be represented by many different words (e.g. "car" and "automobile" refer to the same class of objects). Polysemy refers to the fact that a single word can refer to more than one underlying concept (e.g. "apple" is both a fruit and a computer company). Synonymy results in false negatives and polysemy results in false positives.

Latent semantic indexing is one proposal for addressing this problem. LSI constructs a smaller document matrix that retains only the most important information from the original, by using the Singular Value Decomposition (SVD). Briefly, the SVD of a matrix $D$ is: $USV^T$ where $U$ and $V$ contain orthogonal vectors and $S$ is diagonal (see (Golub and Loan, 1993) for further properties and algorithms). Note that the co-occurrence matrix, $DD^T$, can be written as $US^2U^T$; $U$ contains the eigenvectors of the co-occurrence matrix while the diagonal elements of $S$ (referred to as *singular values*) contain the square roots of their corresponding eigenvalues. The eigenvectors with the largest eigenvalues capture the axes of largest variation in the data.

In LSI, each document is projected into a lower dimensional space $\hat{D} = \hat{S}_k^{-1}\hat{U}_k^T D$ where $\hat{S}_k$ and $\hat{U}_k$ which contain only the largest $k$ singular values and the corresponding eigenvectors, respectively. The resulting document matrix is of smaller size but still provably represents the most variation in the original matrix. Thus, LSI represents documents as linear combinations of orthogonal features. It is hoped that these features represent meaningful underlying "topics" present in the collection. Queries are also projected into this space, so the relevance of documents to a query is $D^T\hat{U}_k\hat{S}_k^{-2}\hat{U}_k^T q$.

This type of dimensionality reduction is very similar to principal components analysis (PCA), which has been used in other domains, including visual object recognition (Turk and Pentland, 1991). In practice, there is some evidence to suggest that LSI can improve retrieval performance; however, it is often the case that LSI improves text retrieval performance by only a small amount or not at all (see (Hull, 1994) and (Deerwester et al., 1990) for a discussion).

## 3   Do Optimal Projections for Retrieval Exist?

Hypotheses abound for the success of LSI, including: i) LSI removes noise from the document set; ii) LSI finds words that are synonyms; iii) LSI finds clusters of documents. Whatever it does, LSI operates without knowledge of the queries that will be presented to the system. We could instead attempt a supervised approach, searching for a matrix $P$ such that $D^T PP^T q$ results in large values for documents in $D$ that are known to be relevant for a particular query, $q$. The choice for the structure of $P$ embodies assumptions about the structure of $D$ and $q$ and what it means for documents and queries to be related.

For example, imagine that we are given a collection of documents, $D$, and queries, $Q$. For each query we are told which documents are relevant. We can use this information to construct an optimal $P$ such that: $D^T PP^T Q \approx R$, where $R_{ij}$ equals 1 if document $i$ is relevant to query $j$, and 0 otherwise.

We find $P$ in two steps. First we find an $X$ minimizing $\|D^T XQ - R\|_F$, where $\|\cdot\|_F$ denotes the Frobenius norm of a matrix[2]. Second, we find $P$ by decomposing $X$ into $PP^T$. Unfortunately, this may not be simple. The matrix $PP^T$ has properties that are not necessarily shared by $X$. In particular, while $PP^T$ is symmetric, there is no guarantee that $X$ will be (in our experiments $X$ is far from symmetric). We can however take SVD of $X = U_x S_x V_x^T$, using matrix $U_x$ to project the documents and $V_x$ to project the queries.

We can now compare LSI's projection axes, $U$ with the optimal $U_x$ computed as above. One measure of comparison is the distribution of documents as projected onto these axes. Figure 2a shows the distribution of Medline documents[3] projected onto the first axis of $U_x$. Notice that there is a large

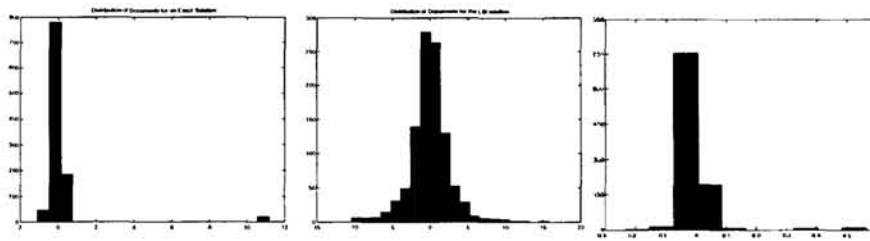

Figure 2: **(A).** The distribution of medline documents projected onto one of the "optimal" axes. The kurtosis of this distribution is 44. **(B).** The distribution of medline documents projected onto one of the LSIaxes. The kurtosis of this distribution is 6.9. **(C).** The distribution of medline documents projected onto one of the ICA axes. The kurtosis of this distribution is 60.

spike near zero, and a well-separated outlier spike. The kurtosis of this distribution is 44. Subsequent axes of $U_x$ result in similar distributions. We might hope that these axes each represent a topic shared by a few documents. Figure 2b shows the distribution of documents projected onto the first LSI axis. This axis yields a distribution with a much lower kurtosis of 6.9 (a normal distribution has kurtosis 3). This induces a distribution that looks nothing like a cluster: there is a smooth continuum of values. Similar distributions result for many of the first 100 axes.

These results suggest that LSI-like approaches may well be searching for projections that are sub-optimal. In the next section, we describe an algorithm designed to find projections that look more like those in Figure 2a than in Figure 2b.

## 4   Topic Centered Representations

There are several problems with the "optimal" approach described in the previous section. Aside from its completely supervised nature, there may be a problem of over-fitting: the number of parameters in $X$ (the number of words squared) can be large compared to the number of documents and queries. It is not clear how to move towards a solution that will likely have low generalization error, our ultimate goal. Further, computing $X$ is expensive, involving several full-rank singular value decompositions.

On the other hand, while we may not be able to take advantage of supervision, it seems reasonable to search for projections like those in Figure 2a. There are several unsupervised techniques we might use. We begin with independent component analysis (Bell and Sejnowski, 1995), a technique that has recently gained popularity. Extensions such as (Amari, Cichocki and Yang, 1996) have made the algorithm more efficient and robust.

### 4.1   What are the Independent Components of Documents?

Figure 2C shows the distribution of Medline documents along one of the ICA axes (kurtosis 60). It is representative of other axes found for that collection, and for other, larger collections.

Like the optimal axes found earlier, this axis also separates documents. This is desirable because it means that the axes are distinguishing groups of (presumably related) documents. Still, we can ask a more interesting question; namely, how do these axes group words? Rather than project our documents onto the ICA space, we can project individual words (this amounts to projecting the identity matrix onto that space) and observe how ICA redistributes them.

Figure 3 shows a typical distribution of all the words along one of the axes found by ICA on the

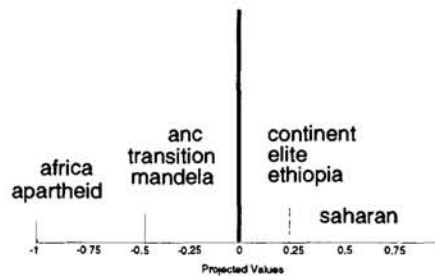

Figure 3: The distribution of words with large magnitude along an ICA axis from the White House collection.

White House collection.[4] ICA induces a highly kurtotic distribution over the words. It is also quite sparse: most words have a value very close to zero. The histogram shows only the words large values, both positive and negative. One group of words is made up of highly-related words; namely, "africa," "apartheid," and "mandela." The other is made up of words that have no obvious relationship to one another. In fact, these words are not directly related, but each co-occurs with different individual words in the first group. For example, "saharan" and "africa" occur together many times, but not in the context of apartheid and South Africa; rather, in documents concerning US policy toward Africa in general. As it so happens, "saharan" acts as a discriminating word for these subtopics.

## 4.2  Topic Centered Representations

It appears that ICA is finding a set of words, $S$, that selects for related documents, $H$, along with another set of words, $T$, whose elements do not select for $H$, but co-occur with elements of $S$. Intuitively, $S$ selects for documents in a general subject area, and $T$ removes a specific subset of those documents, leaving a small set of highly related documents. This suggests a straightforward algorithm to achieve the same goal directly:

```
foreach topic, C^k, you wish to define:
   -Choose a source document d_c from D
   -Let D̂ be the documents of D sorted by similarity to d_c
   -Divide D̂ into into three groups:  those assumed to be relevant,
     those assumed to be completely irrelevant,
     and those assumed to be weakly relevant.
   -Let G^k, B^k, and M^k be the centroid of each respective group
   -Let C^k = f(G^k - B^k) - f(M^k - G^k)
     where f(x) = max(x,0).
```

The three groups of documents are used to drive the discovery of two sets of words. One set selects for documents in a general topic area by finding the set of words that distinguish the relevant documents from documents in general, a form of global clustering. The other set of words distinguish the weakly-related documents from the relevant documents. Assigning them negative weight results in their removal. This leaves only a set of closely related documents. This local clustering approach is similar to an unsupervised version of Rocchio with Query Zoning (Singhal, 1997).

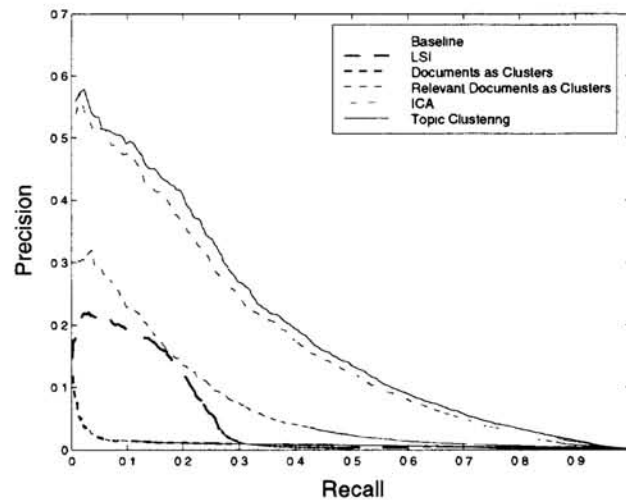

Figure 4: A comparison of different algorithms on the Wall Street Journal

## 5  Experiments

In this section, we show results of experiments with the Wall Street Journal collection. It contains 42,652 documents and 89757 words. Following convention, we measure the success of a text retrieval system using precision-recall curves[5]. Figure 4 illustrates the performance of several algorithms:

1. Baseline: the standard inner product measure, $D^T q$.

2. LSI: Latent Semantic Indexing.

3. ˙ ɔcuments as Clusters: each document is a projection axis. This is equivalent to a modified inner product measure, $D^T D D^T q$.

4. Relevant Documents as Clusters: In order to simulate psuedo-relevance feedback, we use the centroid of the top few documents returned by the $D^T q$ similarity measure.

5. ICA: Independent Component Analysis.

6. Topic Clustering: The algorithm described in Section 4.2.

In this graph, we restrict queries to those that have at least fifty relevant documents. The topic clustering approach and ICA perform best, maintaining higher average precision over all ranges. Unlike smaller collections such as Medline, documents from this collection do not tend to cluster around the queries naturally. As a result, the baseline inner product measure performs poorly. Other clustering techniques that tend to work well on collections such as Medline perform even worse. Finally, LSI does not perform well.

Figure 5 illustrates different approaches on subsets of Wall Street Journal queries. In general, as each query has more and more relevant documents, overall performance improves. In particular, the simple clustering scheme using only relevant documents performs very well. Nonetheless, our approach improves upon this standard technique with minimal additional computation.

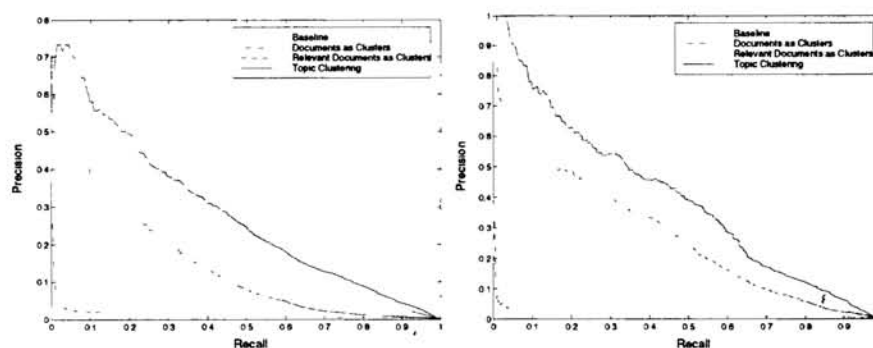

Figure 5: **(A)**. Performance of various clustering techniques for those queries with more than 75 relevant documents. **(B)**. Performance for those queries with more than 100 relevant documents.

## 6   Discussion

We have described typical dimension reduction techniques used in text retrieval and shown that these techniques make strong assumptions about the form of projection axes. We have characterized another set of assumptions and derived an algorithm that enjoys significant computational and space advantages. Further, we have described experiments that suggest that this approach is robust. Finally, much of what we have described here is not specific to text retrieval. Hopefully, similar characterizations will apply to other sparse high-dimensional domains.

## Footnotes

[1]In practice, suffixes are removed and counts are re-weighted by some function of their natural frequency

[2]First find $M$ that minimizes $\|D^T M - R\|_F$. $X$ is the matrix that minimizes $\|XQ - M\|_F$

[3]Medline is a small test collection, consisting of 1033 documents and about 8500 distinct words. We have found similar results for other, larger collections.

[4]The White House collection contains transcripts of press releases and press conferences from 1993. There are 1585 documents and 18675 distinct words.

[5]When asked to return $n$ documents *precision* is the percentage of those which are relavant. *Recall* is the percentage of the total relevant documents which are returned.

## References

Amari, S., Cichocki, A., and Yang, H. (1996). A new learning algorithm for blind source separation. In *Advances in Neural Information Processing Systems*.

Bell, A. and Sejnowski, T. (1995). An information-maximizaton approach to blind source separation and blind deconvolution. *Neural Computation*, 7:1129–1159.

Deerwester, S., Dumais, S. T., Landauer, T. K., Furnas, G. W., and Harshman, R. A. (1990). Indexing by latent semantic analysis. *Journal of the Society for Information Science*, 41(6):391–407.

Frakes, W. B. and Baeza-Yates, R., editors (1992). *Information Retrieval: Data Structures and Algorithms*. Prentice-Hall.

Golub, G. H. and Loan, C. F. V. (1993). *Matrix Computations*. The Johns Hopkins University Press.

Hull, D. (1994). Improving text retrieval for the routing problem using latent semantic indexing. In *Proceedings of the 17th ACM/SIGIR Conference*, pages 282–290.

Kwok, K. L. (1996). A new method of weighting query terms for ad-hoc retrieval. In *Proceedings of the 19th ACM/SIGIR Conference*, pages 187–195.

O'Brien, G. W. (1994). Information management tools for updating an svd-encoded indexing scheme. Technical Report UT-CS-94-259, University of Tennessee.

Sahami, M., Hearst, M., and Saund, E. (1996). Applying the multiple cause mixture model to text categorization. In *Proceedings of the 13th International Machine Learning Conference*.

Salton, G., editor (1971). *The SMART Retrieval System: Experiments in Automatic Document Processing*. Prentice-Hall.

Singhal, A. (1997). Learning routing queries in a query zone. In *Proceedings of the 20th International Conference on Research and Development in Information Retrieval*.

Turk, M. A. and Pentland, A. P. (1991). Face recognition using eigenfaces. In *IEEE Conference on Computer Vision and Pattern Recognition*, pages 586–591.
